# On Reversing Jensen's Inequality

**Tony Jebara**
MIT Media Lab
Cambridge, MA 02139
*jebara@media.mit.edu*

**Alex Pentland**
MIT Media Lab
Cambridge, MA 02139
*sandy@media.mit.edu*

## Abstract

Jensen's inequality is a powerful mathematical tool and one of the workhorses in statistical learning. Its applications therein include the EM algorithm, Bayesian estimation and Bayesian inference. Jensen computes simple lower bounds on otherwise intractable quantities such as products of sums and latent log-likelihoods. This simplification then permits operations like integration and maximization. Quite often (i.e. in discriminative learning) upper bounds are needed as well. We derive and prove an efficient analytic inequality that provides such variational upper bounds. This inequality holds for latent variable mixtures of exponential family distributions and thus spans a wide range of contemporary statistical models. We also discuss applications of the upper bounds including maximum conditional likelihood, large margin discriminative models and conditional Bayesian inference. Convergence, efficiency and prediction results are shown. [1]

## 1 Introduction

Statistical model estimation and inference often require the maximization, evaluation, and integration of complicated mathematical expressions. One approach for simplifying the computations is to find and manipulate variational upper and lower bounds instead of the expressions themselves. A prominent tool for computing such bounds is Jensen's inequality which subsumes many information-theoretic bounds (cf. Cover and Thomas 1996). In maximum likelihood (ML) estimation under incomplete data, Jensen is used to derive an iterative EM algorithm [2]. For graphical models, intractable inference and estimation is performed via variational bounds [7]. Bayesian integration also uses Jensen and EM-like bounds to compute integrals that are otherwise intractable [9].

Recently, however, the learning community has seen the proliferation of conditional or discriminative criteria. These include support vector machines, maximum entropy discrimination distributions [4], and discriminative HMMs [3]. These criteria allocate resources with the given task (classification or regression) in mind, yielding improved performance. In contrast, under canonical ML each density is trained separately to *describe* observations rather than optimize classification or regression. Therefore performance is compromised.

Computationally, what differentiates these criteria from ML is that they not only require Jensen-type lower bounds but may also utilize the corresponding upper bounds. The Jensen bounds only partially simplify their expressions and some intractabilities remain. For instance, latent distributions need to be bounded above and below in a discriminative setting [4] [3]. Metaphorically, discriminative learning requires lower bounds to cluster positive examples and upper bounds to *repel* away from negative ones. We derive these complementary upper bounds [2] which are useful for discriminative classification and regression. These bounds are structurally similar to Jensen bounds, allowing easy migration of ML techniques to discriminative settings.

This paper is organized as follows: We introduce the probabilistic models we will use: mixtures of the exponential family. We then describe some estimation criteria on these models which are intractable. One simplification is to lower bound via Jensen's inequality or EM. The reverse upper bound is then derived. We show implementation and results of the bounds in applications (i.e. conditional maximum likelihood (CML)). Finally, a strict algebraic proof is given to validate the reverse-bound.

## 2 The Exponential Family

We restrict the reverse-Jensen bounds to mixtures of the exponential family (e-family). In practice this class of densities covers a very large portion of contemporary statistical models. Mixtures of the e-family include Gaussians Mixture Models, Multinomials, Poisson, Hidden Markov Models, Sigmoidal Belief Networks, Discrete Bayesian Networks, etc. [1] The e-family has the following form: $P(X|\Theta) = \exp(\mathcal{A}(X) + X^T\Theta - \mathcal{K}(\Theta))$.

| E-Distribution | $\mathcal{A}(X)$ | $\mathcal{K}(\Theta)$ |
|---|---|---|
| Gaussian | $-\frac{1}{2}X^T X - \frac{D}{2}\log(2\pi)$ | $\frac{1}{2}\Theta^T\Theta$ |
| Multinomial | 0 | $\eta\log(1+\sum_d \exp(\Theta_d))$ |

Here, $\mathcal{K}(\Theta)$ is convex in $\Theta$, a multi-dimensional parameter vector. Typically the data vector $X$ is constrained to live in the gradient space of $\mathcal{K}$, i.e. $X \in \frac{\partial}{\partial\Theta}\mathcal{K}(\Theta)$. The e-family has special properties (i.e. conjugates, convexity, linearity, etc.) [1]. The reverse-Jensen bound also exploits these intrinsic properties. The table above lists example $\mathcal{A}$ and $\mathcal{K}$ functions for Gaussian and multinomial distributions. More generally, though, we will deal with mixtures of the e-family (where $m$ represents the incomplete data)[3], i.e.:

$$p(X|\Theta) = \sum_m p(m)p(X,\Theta|m) = \sum_m \alpha_m \exp(\mathcal{A}_m(X_m) + X_m^T\Theta_m - \mathcal{K}_m(\Theta_m))$$

These latent probability distributions need to get maximized, integrated, marginalized, conditioned, etc. to solve various inference, prediction, and parameter estimation tasks. However, such manipulations can be difficult or intractable.

## 3 Conditional and Discriminative Criteria

The combination of ML with EM and Jensen have indeed produced straightforward and monotonically convergent estimation procedures for mixtures of the e-family [2] [1] [7]. However, ML criteria are non-discriminative *modeling* techniques for estimating generative models. Consequently, they suffer when model assumptions are inaccurate.

ML Classifier: $l = -8.0$, $l^c = -1.7$     CML Classifier: $l = -54.7$, $l^c = 0.4$

Figure 1: ML vs. CML (Thick Gaussians represent circles, thin ones represent x's).

For visualization, observe the binary classification[4] problem above. Here, our model incorrectly has 2 Gaussians (identity covariances) per class but the true data is generated from 8 Gaussians. Two solutions are shown, ML and CML. Note the values of joint log-likelihood $l$ and conditional log-likelihood $l^c$. The ML solution performs as well as random chance guessing while CML classifies the data very well. Thus, CML, in estimating a conditional density, propagates the classification task into the estimation criterion.

In such examples, we are given training examples $X_i$ and corresponding binary labels $c_i$ to classify with a latent variable e-family model (mixture of Gaussians). We use $m$ to represent the latent missing variables. The corresponding objective functions log-likelihood $l$ and conditional log-likelihood $l^c$ are:

$$
\begin{aligned}
l &= \sum_i \log \sum_m p(m, c_i, X_i | \Theta) \\
l^c &= \sum_i \log \sum_m p(m, c_i | X_i, \Theta) = \sum_i \left[ \log \sum_m p(m, c_i, X_i | \Theta) - \log \sum_m \sum_c p(m, c, X_i | \Theta) \right]
\end{aligned}
$$

The classification and regression task can be even more powerfully exploited in the case of discriminative (or large-margin) estimation [4] [5]. Here, hard constraints are posed on a *discriminant function* $\mathcal{L}(X|\Theta)$, the ratio of each class' latent likelihood. Prediction of class labels is done via the sign of the function, $\hat{c} = \text{sign}\mathcal{L}(X|\Theta)$.

$$
\mathcal{L}(X|\Theta) = \log \frac{p(X|\Theta_+)}{p(X|\Theta_-)} = \log \sum_m p(m, X|\Theta_+) - \log \sum_m p(m, X|\Theta_-) \tag{1}
$$

In the above log-likelihoods and discriminant functions we note logarithms of sums (latent likelihood is basically a product of sums) which cause intractabilities. For instance, it is difficult to maximize or integrate the above log-sum quantities. Thus, we need to invoke simplifying bounds.

## 4 Jensen and EM Bounds

Recall the definition of Jensen's inequality: $f(E\{\mathcal{X}\}) \geq E\{f(\mathcal{X})\}$ for concave $f$. The log-summations in $l$, $l^c$, and $\mathcal{L}(X|\Theta)$ all involve a concave $f = \log$ around an expectation, i.e. a *log-sum* or probabilistic mixture over latent variables. We apply Jensen as follows:

$$
\log \sum_m p(m, X | \Theta) \geq \sum_m \left[ \frac{p(m, X | \tilde{\Theta})}{\sum_n p(n, X | \tilde{\Theta})} \right] \log \frac{p(m, X | \Theta)}{p(m, X | \tilde{\Theta})} + \log \sum_m p(m, X | \tilde{\Theta})
$$

$$
\log \sum_m \alpha_m \exp(\mathcal{A}_m(X_m) + X_m^T \Theta_m - \mathcal{K}_m(\Theta_m)) \geq \sum_m [h_m] (X_m^T \Theta_m - \mathcal{K}_m(\Theta_m)) + C
$$

Above, we have also expanded the bound in the e-family notation. This forms a variational lower bound on the log-sum which makes tangential contact with it at $\tilde{\Theta}$ and is much easier

to manipulate. Basically, the log-sum becomes a sum of log-exponential family members. There is an additive constant term C and the positive scalar $h_m$ terms (the *responsibilities*) are given by the terms in the square brackets (here, brackets are for grouping terms and are not operators). These quantities are relatively straightforward to compute. We only require *local* evaluations of log-sum values at the current $\Theta$ to compute a *global* lower bound.

If we bound all log-sums in the log-likelihood, we have a lower bound on the objective $l$ which we can maximize easily. Iterating maximization and lower bound computation at the new $\Theta$ produces a local maximum of log-likelihood as in EM. However, applying Jensen on log-sums in $l^c$ and $\mathcal{L}(X|\Theta)$ is not as straightforward. Some terms in these expressions involve *negative* log-sums and so Jensen is actually solving for an *upper bound* on those terms. If we want overall lower and upper bounds on $l^c$ and $\mathcal{L}(X|\Theta)$, we need to compute reverse-Jensen bounds.

## 5 Reverse-Jensen Bounds

It seems strange we can reverse Jensen (i.e. $f(E\{\mathcal{X}\}) \leq E\{f(\mathcal{X})\}$) but it is possible. We need to exploit the convexity of the $\mathcal{K}$ functions in the e-family instead of exploiting the concavity of $f = \log$. However, not only does the reverse-bound have to upper-bound the log-sum, it should also have the same form as the Jensen-bound above, i.e. a sum of log-exponential family terms. That way, upper and lower bounds can be combined homogeneously and ML tools can be quickly adapted to the new bounds. We thus need:

$$\log \sum_m \alpha_m \exp(\mathcal{A}_m(X_m) + X_m^T \Theta_m - \mathcal{K}_m(\Theta_m)) \quad \leq \quad \sum_m -[w_m](Y_m^T \Theta_m - \mathcal{K}_m(\Theta_m)) + k \qquad (2)$$

Here, we give the parameters of the bound directly, refer to the proof at the end of the paper for their algebraic derivation. This bound again makes tangential contact at $\tilde{\Theta}$ yet is an upper bound on the log-sum [5].

$$
\begin{aligned}
k &= \log p(X|\tilde{\Theta}) + \sum_m w_m(Y_m^T \tilde{\Theta}_m - \mathcal{K}_m(\tilde{\Theta}_m)) \\
Y_m &= \frac{h_m}{w_m}\left( \left.\frac{\partial \mathcal{K}(\Theta_m)}{\partial \Theta_m}\right|_{\tilde{\Theta}_m} - X_m \right) + \left.\frac{\partial \mathcal{K}(\Theta_m)}{\partial \Theta_m}\right|_{\tilde{\Theta}_m} \\
w'_m &= \min w'_m \text{ such that } \frac{h_m}{w'_m}\left( \left.\frac{\partial \mathcal{K}(\Theta_m)}{\partial \Theta_m}\right|_{\tilde{\Theta}_m} - X_m \right) + \left.\frac{\partial \mathcal{K}(\Theta_m)}{\partial \Theta_m}\right|_{\tilde{\Theta}_m} \in \frac{\partial \mathcal{K}(\Theta_m)}{\partial \Theta_m} \\
w_m &= [X_m - \mathcal{K}'(\tilde{\Theta}_m)]^T \mathcal{K}''(\tilde{\Theta}_m)^{-1} [X_m - \mathcal{K}'(\tilde{\Theta}_m)] + w'_m
\end{aligned}
$$

This bound effectively reweights ($w_m$) and translates ($Y_m$) incomplete data to obtain complete data. Tighter bounds are possible (i.e. smaller $w_m$) which also depend on the $h_m$ terms (see web page). The first condition requires that the $w'_m$ generate a valid $Y_m$ that lives in the gradient space of the $\mathcal{K}$ functions (a typical e-family constraint). Thus, from *local* computations of the log-sum's values, gradients and Hessians at the current $\tilde{\Theta}$, we can compute *global* upper bounds.

## 6 Applications and Results

In Fig. 2 we plot the bounds for a two-component unidimensional Gaussian mixture model case and a two component binomial (unidimensional multinomial) mixture model. The Jensen-type bounds as well as the reverse-Jensen bounds are shown at various configurations of $\tilde{\Theta}$ and $X$. Jensen bounds are usually tighter but this is inevitable due to the intrinsic shape of the log-sum. In addition to viewing many such 2D visualizations, we computed higher dimensional bounds and sampled them extensively, empirically verifying that the reverse-Jensen bound remained above the log-sum. Below we describe practical uses of this new reverse-bound.

(a) Gaussian Case    (b) Multinomial Case

Figure 2: Jensen (black) and reverse-Jensen (white) bounds on the log-sum (gray).

## 6.1 Conditional Maximum Likelihood

The inequalities above were use to fully lower bound $l^c$ and maximizing the bound iteratively. This is like the CEM algorithm [6] except the new bounds handle the whole e-family (i.e. *generalized* CEM). The synthetic Gaussian mixture model problem problem portrayed in Fig. 1 was implemented. Both ML and CML estimators (with reverse-bounds) were initialized in the same random configuration and maximized. The Gaussians converged as in Fig. 1. CML classification accuracy was 93% while ML obtained 59%. Figure (**A**) depicts the convergence of $l^c$ per iteration under CML (top line) and ML (bottom-line). Similarly, we computed multinomial models for 3-class data as 60 base-pair protein chains in Figure (**B**).

Computationally, utilizing both Jensen and reverse-Jensen bounds for optimizing CML needs double the processing as ML using EM. For example, we estimated 2 classes of mixtures of multinomials (5-way mixture) from 40 10-dimensional data points. In non-optimized Matlab code, ML took 0.57 seconds per epoch while CML took 1.27 seconds due to extra bound computations. Thus, efficiency is close to EM for practical problems. Complexity per epoch roughly scales linearly with sample size, dimensions and number of latent variables.

## 6.2 Conditional Variational Bayesian Inference

In [9], Bayesian integration methods were demonstrated on latent-variable models by invoking Jensen type lower bounds on the integrals of interest. A similar technique can be used to approximate conditional Bayesian integration. Traditionally, we compute the joint Bayesian integral from $(\mathcal{X}, \mathcal{Y})$ data as $p(X, Y) = \int p(X, Y|\Theta)p(\Theta|\mathcal{X}, \mathcal{Y})d\Theta$ and condition it to obtain $p(Y|X)^j$ (the superscript indicates we initially estimated a joint density) . Alternatively, we can compute the conditional Bayesian integral directly. The

corresponding dependency graphs (Fig. 3(b) and (c)) depict the differences between joint and conditional estimation. The conditional Bayesian integral exploits the graph's factorization, to solve $p(Y|X)^c$.

$$p(Y|X)^c = \int p(Y|X,\Theta^c)[p(\Theta^c|\mathcal{X},\mathcal{Y})]d\Theta^c = \int p(Y|X,\Theta^c)\left[\frac{p(\mathcal{Y}|\mathcal{X},\Theta^c)p(\Theta^c)}{p(\mathcal{Y}|\mathcal{X})}\right]d\Theta^c$$

Jensen and reverse-Jensen bound the terms to permit analytic integration. Iterating this process efficiently converges to an approximation of the true integral. We also *exhaustively* solved both Bayesian integrals exactly for a 2 Gaussian mixture model on 4 data points. Fig. 3 shows the data and densities. In Fig. 3(d) joint and conditional estimates are inconsistent under Bayesian integration (i.e. $P(Y|X)^c \neq P(Y|X)^j$).

(a) Data  (b) Conditioned Joint  (c) Direct Conditional  (d) Inconsistency

Figure 3: Conditioned Joint and Conditional Bayesian Estimates

### 6.3   Maximum Entropy Discrimination

Recently, Maximum Entropy Discrimination (MED) was proposed as an alternative criterion for estimating discriminative exponential densities [4] [5] and was shown to subsume SVMs. The technique integrates over discriminant functions like Eq. 1 but this is intractable under latent variable situations. However, if Jensen and reverse-Jensen bounds are used, the required computations can be done. This permits iterative MED solutions to obtain large margin mixture models and mixtures of SVMs (see web page).

## 7   Discussion

We derived and proved an upper bound on the log-sum of e-family distributions that acts as the reverse of the Jensen lower bound. This tool has applications in conditional and discriminative learning for latent variable models. For further results, extensions, etc. see: http://www.media.mit.edu/ ~jebara/bounds.

## 8   Proof

Starting from Eq. 2, we directly compute $k$ and $Y_m$ by ensuring the variational bound makes tangential contact with the log-sum at $\tilde{\Theta}$ (i.e. making their value and gradients equal). Substituting $k$ and $Y_m$ into Eq. 2, we get constraints on $w_m$ via Bregman distances:

$$\sum_m w_m(\mathcal{K}(\Theta_m) - \mathcal{K}(\tilde{\Theta}_m) - (\Theta_m - \tilde{\Theta}_m)^T\mathcal{K}'(\tilde{\Theta}_m)) \geq \log\frac{p(X|\Theta)}{p(X|\tilde{\Theta})} + \sum_m h_m(\Theta_m - \tilde{\Theta}_m)^T(\mathcal{K}'(\tilde{\Theta}_m) - X_m)$$

Define $\mathcal{F}_m(\Theta_m) = \mathcal{K}(\Theta_m) - \mathcal{K}(\tilde{\Theta}_m) - (\Theta_m - \tilde{\Theta}_m)^T\mathcal{K}'(\tilde{\Theta}_m)$. The $\mathcal{F}$ functions are convex and have a minimum (which is zero) at $\tilde{\Theta}_m$. Replace the $\mathcal{K}$ functions with $\mathcal{F}$:

$$\sum_m w_m \mathcal{F}(\Theta_m) \geq \log\frac{\sum_m \exp\{D_m + \Theta_m^T Z_m - \mathcal{F}(\Theta_m)\}}{\sum_m \exp\{D_m + \tilde{\Theta}_m^T Z_m - \mathcal{F}(\tilde{\Theta}_m)\}} - \sum_m h_m(\Theta_m - \tilde{\Theta}_m)^T Z_m$$

Here, $D_m$ are constants and $Z_m := X_m - \mathcal{K}'(\tilde{\Theta}_m)$. Next, define a mapping from these bowl-shaped functions to quadratics:

$$\mathcal{F}_m(\Theta_m) = \mathcal{G}_m(\Phi_m) = \tfrac{1}{2}(\Phi_m - \tilde{\Phi}_m)^T(\Phi_m - \tilde{\Phi}_m)$$

This permits us to rewrite Eq. 2 in terms of $\Phi$:

$$\sum_m w_m \mathcal{G}(\Phi_m) \geq \log \frac{\sum_m \exp\left\{D_m + \Theta_m(\Phi_m)^T Z_m - \mathcal{G}(\Phi_m)\right\}}{\sum_m \exp\{D_m + \tilde{\Theta}_m^T Z_m - \mathcal{G}(\tilde{\Theta}_m)\}} - \sum_m h_m(\Theta_m(\Phi_m) - \tilde{\Theta}_m)^T Z_m \quad (3)$$

Let us find properties of the mapping $\mathcal{F} = \mathcal{G}$. Take 2nd derivatives over $\Phi_m$:

$$\mathcal{K}''(\Theta_m)\frac{\partial \Theta_m}{\partial \Phi_m}\frac{\partial \Theta_m}{\partial \Phi_m}^T + (\mathcal{K}'(\Theta_m) - \mathcal{K}'(\tilde{\Theta}_m))\frac{\partial^2 \Theta_m}{\partial \Phi_m^2} = I$$

Setting $\Theta_m = \tilde{\Theta}_m$ above, we get the following for a family of such mappings: $\left.\frac{\partial \Theta_m}{\partial \Phi_m}\right|_{\tilde{\Theta}_m} = [\mathcal{K}''(\tilde{\Theta}_m)]^{-1/2}$. In an e-family, we can always find a $\Theta_m^\bullet$ such that $X_m = \mathcal{K}'(\Theta_m^\bullet)$. By convexity of $\mathcal{F}$ we create a linear lower bound at $\Theta_m^\bullet$:

$$\mathcal{F}(\Theta_m^\bullet) + (\Theta_m - \Theta_m^\bullet)\left.\frac{\partial \mathcal{F}(\Theta_m)}{\partial \Theta_m}\right|_{\Theta_m^\bullet} \leq \mathcal{F}(\Theta_m) = \mathcal{G}(\Phi_m)$$

Take 2nd derivatives over $\Phi_m$: $\quad \mathcal{F}'(\Theta_m^\bullet)\frac{\partial^2 \Theta_m}{\partial \Phi_m^2} \leq I$ which is rewritten as: $\quad Z_m \frac{\partial^2 \Theta_m}{\partial \Phi_m^2} \leq I$

In Eq. 3, $D_m + \Theta_m(\Phi_m)^T Z_m - \mathcal{G}(\Phi_m)$ is always concave since its Hessian is: $Z_m \frac{\partial^2 \Theta_m}{\partial \Phi_m^2} - I$ which is negative. So, we upper bound these terms by a variational linear bound at $\tilde{\Theta}_m$:

$$\sum_m w_m \mathcal{G}(\Phi_m) \geq \log \frac{\sum_m \exp\left\{D_m' + \Phi_m^T [\mathcal{K}''(\tilde{\Theta}_m)]^{-1/2} Z_m\right\}}{\sum_m \exp\{D_m + \tilde{\Theta}_m^T Z_m - \mathcal{G}(\tilde{\Theta}_m)\}} - \sum_m h_m(\Theta_m(\Phi_m) - \tilde{\Theta}_m)^T Z_m$$

Take 2nd derivatives of both sides with respect to each $\Phi_m$ to obtain (after simplifications):

$$w_m I \geq Z_m \mathcal{K}''(\tilde{\Theta}_m)^{-1} Z_m^T - h_m Z_m \frac{\partial^2 \Theta_m}{\partial \Phi_m^2}$$

If we invoke the constraint on $w_m'$, we can replace $-h_m Z_m \frac{\partial^2 \Theta_m}{\partial \Phi_m^2} \leq w_m' I$. Manipulating, we get the constraint on $w_m$ (as a Loewner ordering here), guaranteeing a global upper bound:

$$w_m I \geq [X_m - \mathcal{K}'(\tilde{\Theta}_m)]\mathcal{K}''(\tilde{\Theta}_m)^{-1}[X_m - \mathcal{K}'(\tilde{\Theta}_m)]^T + w_m' I \qquad \square$$

# 9   Acknowledgments

The authors thank T. Minka, T. Jaakkola and K. Popat for valuable discussions.

## Footnotes

[1]This is the short version of the paper. Please download the long version with tighter bounds, detailed proofs, more results, important extensions and sample matlab code from: http://www.media.mit.edu/ ∼jebara/bounds

[2]A weaker bound for Gaussian mixture regression appears in [6]. Other reverse-bounds are in [8].

[3]Note we use $\Theta$ to denote an aggregate model encompassing all individual $\Theta_m \forall m$.

[4]These derivations extend to multi-class classification and regression as well.

[5]We can also find multinomial bounds on $\alpha$-priors jointly with the $\Theta$ parameters.

# References

[1] Buntine, W. (1994). Operations for learning with graphical models. JAIR 2, 1994.

[2] Dempster, A.P. and Laird, N.M. and Rubin, D.B. (1977). Maximum likelihood from incomplete data via the EM algorithm. *Journal of the Royal Statistical Society*, B39.

[3] Gopalakrishnan, P.S. and Kanevsky, D. and Nadas, A. and Nahamoo, D. (1991). An inequality for rational functions with applications to some statistical estimation problems, IEEE Trans. Information Theory, pp. 107-113, Jan. 1991.

[4] Jaakkola, T. and Meila, M. and Jebara, T. (1999). Maximum entropy discrimination. NIPS 12.

[5] Jebara, T. and Jaakkola, T. (2000). Feature selection and dualities in maximum entropy discrimination. UAI 2000.

[6] Jebara, T. and Pentland, A. (1998). Maximum conditional likelihood via bound maximization and the CEM algorithm. NIPS 11.

[7] Jordan, M. Gharamani, Z. Jaakkola, T. and Saul, L. (1997). An introduction to variational methods for graphical models. *Learning in Graphical Models*, Kluwer Academic.

[8] Pecaric, J.E. and Proschan, F. and Tong, Y.L. (1992). *Convex Functions, Partial Orderings, and Statistical Applications*. Academic Press.

[9] Gharamani, Z. and Beal, M. (1999). Variational Inference for Bayesian Mixture of Factor Analysers, NIPS 12.
